# Efficient Structure Learning of Markov Networks using $L_1$-Regularization

**Su-In Lee    Varun Ganapathi    Daphne Koller**
Department of Computer Science
Stanford University
Stanford, CA 94305-9010
{silee,varung,koller}@cs.stanford.edu

## Abstract

Markov networks are commonly used in a wide variety of applications, ranging from computer vision, to natural language, to computational biology. In most current applications, even those that rely heavily on learned models, the structure of the Markov network is constructed by hand, due to the lack of effective algorithms for learning Markov network structure from data. In this paper, we provide a computationally efficient method for learning Markov network structure from data. Our method is based on the use of $L_1$ regularization on the weights of the log-linear model, which has the effect of biasing the model towards solutions where many of the parameters are zero. This formulation converts the Markov network learning problem into a convex optimization problem in a continuous space, which can be solved using efficient gradient methods. A key issue in this setting is the (unavoidable) use of approximate inference, which can lead to errors in the gradient computation when the network structure is dense. Thus, we explore the use of different feature introduction schemes and compare their performance. We provide results for our method on synthetic data, and on two real world data sets: pixel values in the MNIST data, and genetic sequence variations in the human HapMap data. We show that our $L_1$-based method achieves considerably higher generalization performance than the more standard $L_2$-based method (a Gaussian parameter prior) or pure maximum-likelihood learning. We also show that we can learn MRF network structure at a computational cost that is not much greater than learning parameters alone, demonstrating the existence of a feasible method for this important problem.

## 1  Introduction

Undirected graphical models, such as Markov networks or log-linear models, have been used in an ever-growing variety of applications, including computer vision, natural language, computational biology, and more. However, as this modeling framework is used in increasingly more complex and less well-understood domains, the problem of selecting from the exponentially large space of possible network structures becomes of great importance. Including all of the possibly relevant interactions in the model generally leads to overfitting, and can also lead to difficulties in running inference over the network. Moreover, learning a "good" structure can be an important task in its own right, as it can provide insight about the underlying structure in the domain.

Unfortunately, the problem of learning Markov networks remains a challenge. The key difficulty is that the maximum likelihood (ML) parameters of these networks have no analytic closed form; finding these parameters requires an iterative procedure (such as conjugate gradient [15] or BFGS [5]), where each iteration runs inference over the current model. This type of procedure is computationally expensive even for models where inference is tractable. The problem of structure learning is considerably harder. The dominant type of solution to this problem uses greedy local heuristic search, which incrementally modifies the model by adding and possibly deleting features.

One approach [6, 14] adds features so as to greedily improve the model likelihood; once a feature is added, it is never removed. As the feature addition step is heuristic and greedy, this can lead to the inclusion of unnecessary features, and thereby to overly complex structures and overfitting. An alternative approach [1, 7] explicitly searches over the space of low-treewidth models, but the utility of such models in practice is unclear; indeed, hand-designed models for real-world problems generally do not have low tree-width. Moreover, in all of the greedy heuristic search methods, the learned network is (at best) a local optimum of a penalized likelihood score.

In this paper, we propose a different approach for learning the structure of a log-linear graphical model (or a Markov network). Rather than viewing it as a combinatorial search problem, we embed the structure selection step within the problem of parameter estimation: features that have weight zero (in log-space) are degenerate, and do not induce dependencies between the variables they involve. To appropriately bias the model towards sparsity, we use a technique that has become increasingly popular in the context of supervised learning, in problems that involve a large number of features, many of which may be irrelevant. It has been long known [21] that using $L_1$-regularization over the model parameters — optimizing a joint objective that trades off fit to data with the sum of the absolute values of the parameters — tends to lead to sparse models, where many weights have value 0. More recently, Dudik et al. (2004) showed that density estimation in log-linear models using $L_1$-regularized likelihood has sample complexity that grows only logarithmically in the number of features of the log-linear model; Ng (2004) shows a similar result for $L_1$-regularized logistic regression. These results show that this approach is useful for selecting the relevant features from a large number of irrelevant ones. Other recent work proposes effective algorithms for $L_1$-regularized generalized linear models (e.g., [18, 10, 9]), support vector machines (e.g., [25]), and feature selection in log-linear models encoding natural language grammars [19].

Surprisingly, the use of $L_1$-regularization has not been proposed for the purpose of structure learning in general Markov networks. In this paper, we explore this approach, and discuss issues that are important to its effective application to large problems. A key point is that, for a given $L_1$-based model score and given candidate feature set $\mathcal{F}$, we have a fixed convex optimization problem that admits a unique optimal solution. Due to the properties of the $L_1$ score, in this solution, many features will have weight 0, generally leading to a sparse network structure. However, it is generally impractical to simply initialize the model to include all possible features: exact inference in such a model is almost invariably intractable, and approximate inference methods such as loopy belief propagation [17] are likely to give highly inaccurate estimates of the gradient, leading to poorly learned models. Thus, we propose an algorithm schema that gradually introduces features into the model, and lets the $L_1$-regularization scheme eliminate them via the optimization process. We explore the use of different approaches for feature introduction, one based on the *gain-based method* of Della Pietra, Della Pietra and Lafferty [6] and one on the *grafting method* of Perkins, Lacker and Theiler [18]. We provide a sound termination condition for the algorithm based on the criterion proposed by Perkins et al. [18]; given correct estimates of the gradient, this algorithm is guaranteed to terminate only at the unique global optimum, for any reasonable feature introduction method.

We test our method on synthetic data generated from known MRFs and on two real-world tasks: modeling the joint distribution of pixel values in the MNIST data [12], and modeling the joint distribution of genetic sequence variations — *single-nucleotide polymorphisms (SNPs)* — in the human HapMap data [3]. Our results show that $L_1$-regularization out-performs other approaches, and provides an effective method for learning MRF structure even in large, complex domains.

## 2 Preliminaries

We focus our presentation on the framework of log-linear models, which forms a convenient basis for a discussion of learning. Let $\mathcal{X} = \{X_1, \ldots, X_n\}$ be a set of discrete-valued random variables. A *log-linear model* is a compact representation of a probability distribution over assignments to $\mathcal{X}$. The log-linear model is defined in terms of a set of feature functions $f_k(\boldsymbol{X}_k)$, each of which is a function that defines a numerical value for each assignment $\boldsymbol{x}_k$ to some subset $\boldsymbol{X}_k \subset \mathcal{X}$. Given a set of feature functions $F = \{f_k\}$, the parameters of the log-linear model are weights $\boldsymbol{\theta} = \{\theta_k : f_k \in F\}$. The overall distribution is then defined as: $P_{\boldsymbol{\theta}}(\boldsymbol{x}) = \frac{1}{Z(\boldsymbol{\theta})} \exp(\sum_{f_k \in F} \theta_k f_k(\boldsymbol{x}_k))$, where $\boldsymbol{x}_k$ is the assignment to $\boldsymbol{X}_k$ within $\boldsymbol{x}$, and $Z(\boldsymbol{\theta})$ is the *partition function* that ensures that the distribution is normalized (so that all entries sum to 1). Note that this definition of features encompasses both "standard" features that relate unobserved network variables (e.g., the part of speech of a word in a sentence) to observed elements in the data (e.g., the word itself), and structural features that encode the interaction between hidden variables in the model. A log-linear model induces a *Markov*

*network* over $\mathcal{X}$, where there is an edge between every pair of variables $X_i, X_j$ that appear together in some feature $f_k(\boldsymbol{X}_k)$ ($X_i, X_j \in \boldsymbol{X}_k$). The clique potentials are constructed from the log-linear features in the obvious way. Conversely, every Markov network can be encoded as a log-linear model by defining a feature which is an indicator function for every assignment of variables $\boldsymbol{x}_c$ to a clique $\boldsymbol{X}_c$. The mapping to Markov networks is useful, as most inference algorithms, such as belief propagation [17, 16], operate on the graph structure of the Markov network.

The standard learning problem for MRFs is formulated as follows. We are given a set of IID training instances $\mathcal{D} = \{\boldsymbol{x}[1], \ldots, \boldsymbol{x}[M]\}$, each consisting of a full assignment to the variables in $\mathcal{X}$. Our goal is to output a log-linear model $\mathcal{M}$ over $\mathcal{X}$, which consists of a set of features $F$ and a parameter vector $\boldsymbol{\theta}$ that specifies a weight for each $f_k \in F$.

The log-likelihood function $\log P(\mathcal{D} \mid \mathcal{M})$ has the following form:

$$\ell(\mathcal{M} : \mathcal{D}) = \sum_{f_k \in F} \theta_k f_k(\mathcal{D}) - M \log Z(\boldsymbol{\theta}) = \boldsymbol{\theta}^\top \mathbf{f}(\mathcal{D}) - M \log Z(\boldsymbol{\theta}), \qquad (1)$$

where $f_k(\mathcal{D}) = \sum_{m=1}^{M} f_k(\boldsymbol{x}_k[m])$ is the sum of the feature values over the entire data set, $\mathbf{f}(\mathcal{D})$ is the vector where all of these aggregate features have been arranged in the same order as the parameter vector, and $\boldsymbol{\theta}^\top \mathbf{f}(\mathcal{D})$ is a vector dot-product operation. There is no closed-form solution for the parameters that maximize Eq. (1), but the objective is concave, and can therefore be optimized using numerical optimization procedures such as conjugate gradient [15] or BFGS [5]. The gradient of the log-likelihood is:

$$\frac{\partial \ell(\mathcal{M} : \mathcal{D})}{\partial \theta_k} = f_k(\mathcal{D}) - M \boldsymbol{E}_{\boldsymbol{x} \sim P_{\boldsymbol{\theta}}}[f_k(\boldsymbol{x})]. \qquad (2)$$

This expression has a particularly intuitive form: the gradient attempts to make the feature counts in the empirical data equal to their expected counts relative to the learned model. Note that, to compute the expected feature counts, we must perform inference relative to the current model. This inference step must be performed at every iteration of the gradient process.

## 3 $L_1$-Regularized Structure Learning

We formulate our structure learning problem as follows. We assume that there is a (possibly very large) set of features $\mathcal{F}$, from which we wish to select a subset $F \subseteq \mathcal{F}$ for inclusion in the model. This problem is generally solved using a heuristic search over the combinatorial space of possible feature subsets. Our approach addresses it as a search over the possible parameter vectors $\boldsymbol{\theta} \in I\!\!R^{|\mathcal{F}|}$.

Specifically, rather than optimizing the log-likelihood itself, we introduce a *Laplacian* parameter prior for each feature $f_k$ takes the form $P(\theta_k) = \beta_k/2 \cdot \exp(-\beta_k |\theta_k|)$. We define $P(\boldsymbol{\theta}) = \prod_k P(\theta_k)$. We now optimize the posterior probability $P(\mathcal{D}, \boldsymbol{\theta}) = P(\mathcal{D} \mid \boldsymbol{\theta}) P(\boldsymbol{\theta})$ rather than the likelihood. Taking the logarithm and eliminating constant terms, we obtain the following objective:

$$\max_{\boldsymbol{\theta}} [\boldsymbol{\theta}^\top \mathbf{f}(\mathcal{D}) - M \log Z(\boldsymbol{\theta}) - \sum_k \beta_k |\theta_k|]. \qquad (3)$$

In most cases, the same prior is used for all features, so we have $\beta_k = \beta$ for all $k$. This objective is convex, and can be optimized efficiently using methods such as conjugate gradient or BFGS, although care needs to be taken with the discontinuity of the derivative at $0$. Thus, in principle, we can simply optimize this objective to obtain its globally optimal parameter assignment.

The objective of Eq. (3) should be contrasted with the one obtained for the more standard parameter prior used for log-linear models: the mean-zero Gaussian prior $P(\theta_k) \propto \exp(-\theta_k^2/2\sigma^2)$. The gaussian prior induces a regularization term that is quadratic in $\theta_k$, which penalizes large features much more than smaller ones. Conversely, $L_1$-regularization still penalizes small terms strongly, thereby forcing parameters to $0$. Overall, it is known that, empirically, optimizing an $L_1$-regularized objective leads to a sparse representation with a relative small number of non-zero parameters.

Aside from this intuitive argument, recent theoretical results also provide a formal justification for the use of $L_1$-regularization over other approaches: The analysis of Dudik et al. (2004) and Ng (2004) suggests that this form of regularization is effective at identifying relevant features even with a relatively small number of samples. Building directly on the results of Dudik et al. (2004), we can show the following result:

**Corollary 3.1:** *Let $\mathcal{X} = \{X_1, \ldots, X_n\}$ be a set of variables each of domain size $d$, and $P^*(\mathcal{X})$ be a distribution. Let $\mathcal{F}$ be the set of indicator features over all subsets of variables $\boldsymbol{X} \subset \mathcal{X}$ of*

*cardinality at most c, and $\delta, \epsilon, B > 0$. Let be the parameterization over $\mathcal{F}$ that optimizes*

$$\boldsymbol{\theta}_B^* = \arg \max_{\boldsymbol{\theta} \,:\, \|\boldsymbol{\theta}\| \leq B} \boldsymbol{E}_{\xi \sim P^*} \left[ \ell(\xi : \hat{\boldsymbol{\theta}}_B^*) \right].$$

*For a data set $\mathcal{D}$, let $\hat{\boldsymbol{\theta}}_{\mathcal{D}}$ be the assignment that optimizes Eq. (3), for regularization parameter $\beta_k = \beta = \sqrt{c \ln(2nd/\delta)/(2m)}$ for all $k$. Then with probability at least $1 - \delta$, for a data set $\mathcal{D}$ of IID instances from $P^*$ of size*

$$m \geq 2cB^2 \frac{1}{\epsilon^2} \ln \left( \frac{2nd}{\delta} \right).$$

*we have that:*

$$\boldsymbol{E}_{\xi \sim P^*} \left[ \ell(\xi : \hat{\boldsymbol{\theta}}_{\mathcal{D}}) \right] \geq \boldsymbol{E}_{\xi \sim P^*} [\ell(\xi : \boldsymbol{\theta}_B^*)] - \epsilon.$$

In words, using the $L_1$-regularized log-likelihood objective, we can learn a Markov network with a maximal clique size $c$, whose expected log-likelihood relative to the true underlying distribution is at most $\epsilon$ worse than the log-likelihood of the optimal Markov network in this class whose $L_1$-norm is at most $B$. The number of samples required grows logarithmically in the number of nodes in the network, and polynomially in $B$. The dependence on $B$ is quite natural, indicating that more samples are required to learn networks containing more "strong" interactions. Note, however, that if we bound the magnitude of each potential in the Markov network, then $B = O((nd)^c)$, so that a polynomial number of data instances suffices.

## 4  Incremental Feature Introduction

The above discussion implicitly assumed that we can find the global optimum of Eq. (3) by simple convex optimization. However, we cannot simply include all of the features in the model in advance, and use only parameter optimization to prune away the irrelevant ones. Recall that computing the gradient requires performing inference in the resulting model. If we have too many features, the model may be too densely connected to allow effective inference. Even approximate inference algorithms, such as belief propagation, tend to degrade as the density of the network increases; for example, BP algorithms are less likely to converge, and the answers they return are typically much less accurate. Thus, our approach also contains a feature introduction component, which gradually selects features to add into the model, allowing the optimization process to search for the optimal values for their parameters. More precisely, our algorithm maintains a set of *active features* $F \subseteq \mathcal{F}$. An inactive feature $f_k$ has its parameter $\theta_k$ set to 0; the parameters of active features are free to be changed when optimizing the objective Eq. (3).

In addition to various simple baseline methods, we explore two feature introduction methods, both of which are greedy and myopic, in that they compute some heuristic estimate of the likely benefit to be gained from introducing a single feature into the active set.

The *grafting* procedure of Perkins et al. [18], which was developed for feature selection in standard classification tasks, selects features based on the gradient of these parameters: We first optimize the objective relative to the current active features $F$ and their weights, so that, at convergence, the gradient relative to these features is zero. Then, for each inactive feature $f$, we compute the partial derivative of the objective Eq. (3) relative to $\theta_f$, and select the one whose gradient is largest.

A more conservative estimate is obtained from the *gain-based* method of Della Pietra et al. [6]. This method was designed for the log-likelihood objective. It begins by optimizing the parameters relative to the current active set $F$. Then, for each inactive feature $f$, it computes the log-likelihood gain of adding that feature, assuming that we could optimize its feature weight arbitrarily, but that the weights of all other features are held constant. It then introduces the feature with the greatest gain. Della Pietra et al. show that the gain is a concave objective that can be computed efficiently using a one-dimensional line search. For the restricted case of binary-valued features, they provide a closed-form solution for the gain. Our task is to compute not the optimal gain in log-likelihood, but rather the optimal gain of Eq. (3). It is not difficult to see that the gain in this objective, which differs from the log-likelihood in only a linear term, is also a concave function that can be optimized using line search. Moreover, for the case of binary-valued features, we can also provide a closed-form solution for the gain. The change in the objective function for introducing a feature $f_k$ is:

$$\Delta_{LI} = \theta_k f_k(\mathcal{D}) - \beta \|\theta_k\| - M \log[\exp(\theta_k) P_{\boldsymbol{\theta}}(f_k) + P_{\boldsymbol{\theta}}(\neg f_k)],$$

where $M$ is the number of training instances. If we take the derivative of the above function and set it to zero, we also get a closed form solution:

$$\theta_k = \log \left( \frac{(f_k(\mathcal{D}) - \beta \mathrm{sign}(\theta_k)) P_{\boldsymbol{\theta}}(\neg f_k)}{(M - f_k(\mathcal{D}) + \beta \mathrm{sign}(\theta_k)) P_{\boldsymbol{\theta}}(f_k)} \right).$$

Both methods are heuristic, in that they consider only the potential gain of adding a single feature in isolation, assuming all other weights are held constant. However, the grafting method is more optimistic, in that it estimates the value of adding a single feature via the slope of adding it, whereas the gain-based approach also considers, intuitively, how far one can go in that direction before the gain "peaks out". The gain based heuristic is, in fact, a lower bound on the actual gain obtained from adding this feature, while allowing the other features to also adapt. Overall, the gain-based heuristic provides a better estimate of the value of adding the feature, albeit at slightly greater computational cost (except in the limited cases where a closed-form solution can be found).

As observed by Perkins *et al.* [18], the use of the $L_1$-regularized objective also provides us with a sound stopping criterion for any incremental feature-induction algorithm. If we have that, for every inactive $f_k \notin F$, the gradient of the log-likelihood $|\frac{\partial \ell(\mathcal{M}:\mathcal{D})}{\partial \theta_k}| \leq \beta$, then the gradient of the objective in any direction is non-positive, and the objective is at a stationary point. Importantly, as the overall objective is a concave function, it has a unique global maximum. Hence, once the termination condition is achieved, we are guaranteed that we are at the local maximum, *regardless of the feature introduction method used*. Thus, assuming the algorithm is run until the convergence criterion is satisfied, there is no impact of the feature introduction heuristic on the final outcome, but only on the computational complexity.

Finally, constantly evaluating all of the non-active candidate features can be computationally prohibitive when many features are possible. Even in pairwise Markov networks, when the number of nodes is large, a quadratic number of candidate edges can become unmanageable. In this case, we must generally pre-select a smaller set of candidate features, and ignore the others entirely. One very natural method for pre-selecting edges is to train an $L_1$-regularized logistic regression classifier for each variable given all of the others, and then use only the edges that are used in these individual classifiers. This approach is similar to the work of Wainwright et al. [22] (done in parallel with our work), who proposed the use of $L_1$-regularized pseudo-likelihood for asymptotically learning a Markov network structure.

## 5    The Use of Approximate Inference

All of the steps in the above algorithm rely on the use of inference for computing key quantities: The gradient is needed for the parameter optimization, for the grafting method, and for the termination condition, and the expression for the gradient requires the computation of marginal probabilities relative to our current model. Similarly, the computation of the gain also requires inference. As we discussed above, in most of the networks that are useful models for real applications, exact inference is intractable. Therefore, we must resort to approximate inference, which results in errors in the gradient. While many approximate inference methods have been proposed, one of the most commonly used is the general class of *loopy belief propagation (BP)* algorithms [17, 16, 24]. The use of an approximate inference algorithm such as BP raises several important points.

One important question issue relates to the computation of the gradient or the gain for features that are currently inactive. The belief propagation algorithm, when executed on a particular network with a set of active features $F$, creates a cluster for every subset of variables $\boldsymbol{X}_k$ that appear as the scope of a feature $f_k(\boldsymbol{X}_k)$. The output of the BP inference process is a set of marginal probabilities over all of the clusters; thus, it returns the necessary information for computing the expected sufficient statistics in the gradient of the objective (see Eq. (2)). However, for features $f_k(\boldsymbol{X}_k)$ that are currently inactive, there is no corresponding cluster in the induced Markov network, and hence, in most cases, the necessary marginal probabilities over $\boldsymbol{X}_k$ will not be computed by BP. We can approximate this marginal probability by extracting a subtree of the calibrated loopy graph that contains all of the variables in $\boldsymbol{X}_k$. At convergence of the BP algorithm, every subtree of the loopy graph is calibrated, in that all of the belief potentials must agree [23]. Thus, we can view the subtree as a calibrated clique tree, and use standard dynamic programming methods over the tree (see, e.g.. [4]) to extract an approximate joint distribution over $\boldsymbol{X}_k$. We note that this computation is exact for tree-structured cluster graphs, but approximate otherwise, and that the choice of tree is not obvious, and affects the accuracy of the answers.

A second key issue is that the performance of BP algorithms generally degrades significantly as the density of the network increases: they are less likely to converge, and the answers they return are typically much less accurate. Moreover, non-convergence of the inference is more common when the network parameters are allowed to take larger, more extreme values; see, for example, [20, 11, 13] for some theoretical results supporting this empirical phenomenon. Thus, it is important to keep the model amenable to approximate inference, and thereby continue to improve, for as long as possible. This observation has two important consequences. First, while different feature introduction schemes achieve the same results when using exact inference, their outcomes can vary greatly when using approximate inference, due to differences in the structure of the networks arising during the learning process. Thus, as we shall see, better feature introduction methods, which introduce the more relevant features first, work much better in practice. Second, in order to keep the inference feasible for as long as possible, we utilize an annealing schedule for the regularization parameter $\beta$, beginning with large values of $\beta$, leading to greater sparsification of the structure, and then gradually reducing $\beta$, allowing additional (weaker) features to be introduced. This method allows a greater part of the learning to be executed with a more robust model.

## 6   Results

In our experiments, we focus on binary *pairwise Markov networks*, where each feature function is an indicator function for a certain assignment to a pair of nodes. As computing the exact log-likelihood is intractable in most networks, we use the *conditional marginal log-likelihood (CMLL)* as our evaluation metric on the learned network. To calculate CMLL, we first divide the variables into two groups: $\mathbf{X}_{\text{hidden}}$ and $\mathbf{X}_{\text{observed}}$. Then, for any test instance $\mathbf{X}[m]$, we compute $CMLL(X[m]) = \sum_{X_h \in \mathbf{X}_{\text{hidden}}[m]} \log P(X_h|\mathbf{X}_{\text{observed}}[m])$. In practice, we divide the variables into four groups and calculate the average CMLL when observing only one group and hiding the rest. Note that the CMLL is defined only with respect to the marginals (but not the global partition function $Z(\boldsymbol{\theta})$), which are empirically thought to be more accurate.

We considered three feature induction schemes: (a) **Gain**: based on the estimated change of gain, (b) **Grad**: using grafting and (c) **Simple**: based on pairwise similarity. Under the Simple scheme, the score of a pairwise feature between $X_i$ and $X_j$ is the mutual information between $X_i$ and $X_j$. For each scheme, we varied the regularization method: (a) **None**: no regularization, (b) **L1**: $L_1$ regularization and (c) **L2**: $L_2$ regularization. We note that Gain and Grad performed similarly for L1 and None. Moreover, we used only Grad for L2, because $L_2$ regularization does not admit a closed form solution for the approximate gain.

**Experiments on Synthetically Generated Data.**    We generated synthetic data through Gibbs sampling on a synthetic network. A network structure with $N$ nodes was generated by treating each possible edge as a Bernoulli random variable and sampling the edges. We chose the parameter of Bernoulli distribution so that each node had $K$ neighbors on average. In order to analyze the dependence of the performance on the size and connectivity of a network, we varied $N$ and $K$.

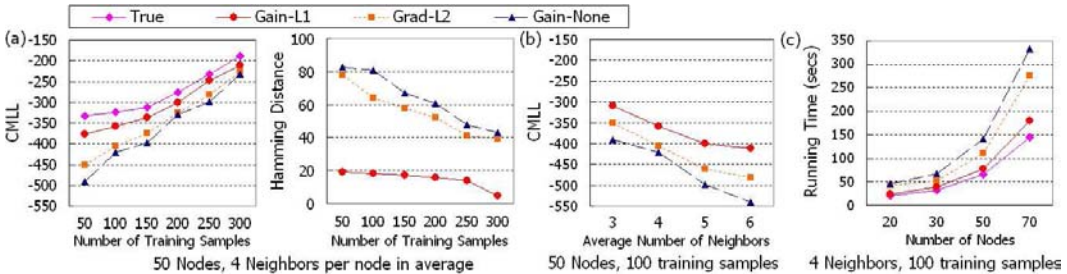

Figure 1: Results from the experiments on the synthetic data (See text for details.)

We compare our algorithm using $L_1$ regularization against no regularization and $L_2$ regularization in three different ways. Figure 1 summarizes our results on this data sets, and includes information about the synthetic networks used for each experiment. The method labeled 'True' simply learns the parameters given the true model. In Figure 1(a), we measure performance using CMLL and reconstruction error as the number of training examples increases. As expected, L1 produces the biggest improvement when the number of training instances is small, whereas L2 and None are more prone to overfitting. This effect is much more pronounced when measuring the Hamming distance, the number of disagreeing edges between the learned structure and the true structure. The figure shows that L2 and None learn many spurious edges. Not surprisingly, L1 shows sparser distribution

on the weights, thereby it has smaller number of edges with non-negligible weights; the structures from None and L2 tend to have many edges with small values. In Figure 1(b), we plot performance as a function of the density of the synthetic network. As the synthetic network gets denser, L1 increasingly outperforms the other algorithms. This may be because as the graph gets more dense, each node is indirectly correlated with more other nodes. Therefore, the feature induction algorithm is more likely to introduce an spurious edge, which L1 may later remove, whereas None and L2 do not. In Figure 1(c), we measure the wall-clock time as a function of the size of the synthetic network. Figure 1(c) shows that the computational cost of learning the structure of the network using Gain-L1 not much more than that of learning the parameters alone. Moreover, L1 increasingly outperforms other regularization methods as the number of nodes increases.

**Experiments on MNIST Digit Dataset.** Moving to real data, we applied our algorithm to hand-written digits. The MNIST training set consists of $32 \times 32$ binary images of handwritten digits. In order to speed up inference and learning, we resized the image to $16 \times 16$. We trained the model where each pixel is a variable for each digit separately, using a training set consisting of 189–195 images per digit. For each digit, we used 50 images as training instances and the remainder as test instances.

Figure 2(a) compares CMLL of the different methods. To save space, we show the digits on which the relative difference in performance of L1 compared to the next best competitor is the lowest (digit 5) and highest (digit 0), as well as the average performance. As mentioned earlier, the performance

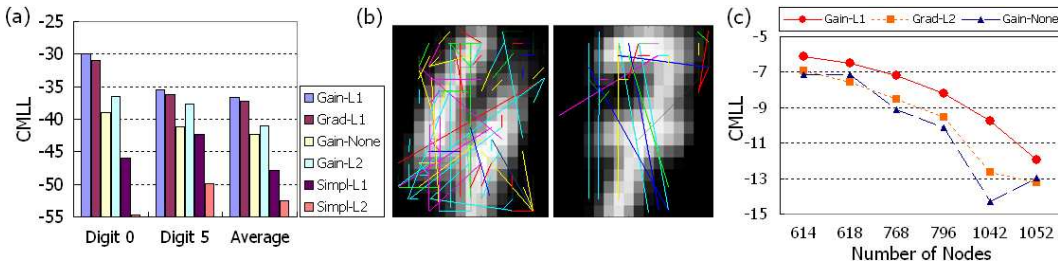

Figure 2: Results from the experiments on MNIST dataset

of the regularized algorithm should be insensitive to the feature induction method, assuming inference is exact. However, in practice, because inference is approximate, an induction algorithm that introduces spurious features will affect the quality of inference, and therefore the performance of the algorithm. This effect is substantiated by the poor performance of the Simple-L1 and Simple-L2 methods that introduce features based on mutual information rather than gradient (Grad-) or approximate gain (Gain-). Nevertheless L1 still outperforms None and L2, regardless the feature induction algorithm with which it is paired. Figure 2(b) shows a visualization of the MRF learned when modeling digits 4 and 7. Of course, one would expect many short-range interactions, such as the associativity between neighboring pixels, and the algorithm does indeed capture these relationships. (They are not shown in the graph to simplify the analysis of the relationships.) Interestingly, the algorithm picks up long-range interactions, which presumably allow the algorithm to model the variations in the size and shape of hand-written digits.

**Experiments on Human Genetic Variation Data.** The Human HapMap data set [1] represents the genetic variation over human individuals. Six data sets contain the genotype values over 614-1,052 genetic markers (SNPs) from 120 individuals. For each data set, we learned the structure of the Markov network whose nodes are binary valued SNPs such that it captures the structure of the human genetic variation. Figure 2(c) compares CMLLs among three methods for these data sets. For all data sets, L1 shows better performance than L2 and None.

## 7 Discussion and Future Work

We have presented a simple and effective method for learning the structure of Markov networks. We view the structure learning problem as an $L_1$-regularized parameter estimation task, allowing it to be solved using convex optimization techniques. We show that the computational cost of our method is not considerably greater than pure parameter estimation for a fixed structure, suggesting that MRF structure learning is a feasible option for many applications.

There are some important directions in which our work can be extended. Currently, our method handles each feature in the log-linear model independently, with no attempt to bias the learning towards sparsity *in the structure of the induced Markov network.* We can extend our approach to introduce such a bias by using a variant of $L_1$ regularization that penalizes blocks of parameters together, such as the block-$L_1$-norm of [2].

From a theoretical perspective, it would be interesting to show that, at the large sample limit, redundant features are eventually eliminated, so that the learning eventually converges to a minimal structure consistent with the underlying distribution. Similar results were shown by Donoho [8], and can perhaps be adapted to this case.

A key limiting factor in MRF learning, and in our approach, is the fact that it requires inference over the model. While our experiments suggest that approximate inference is a viable solution, as the network structure becomes dense, its performance does degrade, especially as the approximate gradient does not always move the parameters to 0, diminishing the sparsifying effect of the $L_1$ regularization, and rendering the inference even less precise. It would be interesting to explore inference methods whose goal is correctly estimating the direction (even if not the magnitude) of the gradient.

Finally, it would be interesting to explore the viability of the learned network structures in real-world applications, both for density estimation and for knowledge discovery, for example, in the context of the HapMap data.

## Footnotes

[1]The Human HapMap data are available at: `http://www.hapmap.org`.

## References

[1] F. Bach and M. Jordan. Thin junction trees. In *NIPS 14*, 2002.

[2] F.R. Bach, G.R.G. Lanckriet, and M.I. Jordan. Multiple kernel learning, conic duality, and the smo algorithm, 2004.

[3] The International HapMap Consortium. The international hapmap project. *Nature*, 426:789–796, 2003.

[4] Robert G. Cowell and David J. Spiegelhalter. *Probabilistic Networks and Expert Systems*. Springer-Verlag New York, Inc., Secaucus, NJ, USA, 1999.

[5] H. Daumé III. Notes on CG and LM-BFGS optimization of logistic regression. August 2004.

[6] S. Della Pietra, V.J. Della Pietra, and J.D. Lafferty. Inducing features of random fields. *IEEE Transactions on Pattern Analysis and Machine Intelligence*, 19(4):380–393, 1997.

[7] A. Deshpande, M.N. Garofalakis, and M.I. Jordan. Efficient stepwise selection in decomposable models. In *Proc. UAI*, pages 128–135, 2001.

[8] D. Donoho and X. Huo. Uncertainty principles and ideal atomic decomposition, 1999.

[9] A. Genkin, D. D. Lewis, and D. Madigan. Large-scale bayesian logistic regression for text categorization. 2004.

[10] J. Goodman. Exponential priors for maximum entropy models. In *North American ACL*, 2005.

[11] Alexander T. Ihler, John W. Fischer III, and Alan S. Willsky. Loopy belief propagation: Convergence and effects of message errors. *J. Mach. Learn. Res.*, 6:905–936, 2005.

[12] Y. LeCun, L. Bottou, Y. Bengio, and P. Haffner. Gradient-based learning applied to document recognition. *Proceedings of the IEEE*, 86(11):2278–2324, November 1998.

[13] Martijn A. R. Leisink and Hilbert J. Kappen. General lower bounds based on computer generated higher order expansions. In *UAI*, pages 293–300, 2002.

[14] A. McCallum. Efficiently inducing features of conditional random fields. In *Proc. UAI*, 2003.

[15] Thomas P. Minka. Algorithms for maximum-likelihood logistic regression. 2001.

[16] K. P. Murphy, Y. Weiss, and M. I. Jordan. Loopy belief propagation for approximate inference: an empirical study. pages 467–475, 1999.

[17] J. Pearl. *Probabilistic Reasoning in Intelligent Systems*. Morgan Kaufmann, 1988.

[18] S. Perkins, K. Lacker, and J. Theiler. Grafting: Fast, incremental feature selection by gradient descent in function space. 3(2003):1333–1356, 2003.

[19] Stefan Riezler and Alexander Vasserman. Incremental feature selection and l1 regularization for relaxed maximum-entropy modeling. In *Proceedings of EMNLP 2004*.

[20] Sekhar Tatikonda and Michael I. Jordan. Loopy belief propogation and gibbs measures. In *UAI*, pages 493–500, 2002.

[21] R Tibshirani. Regression shrinkage and selection via the lasso. *J. Royal. Statist. Soc B*, 1996.

[22] Martin J. Wainwright, Pradeep Ravikumar, and Lafferty. Inferring graphical model structure using $\ell_1$-regularized pseudo-likelihood. In *Advances in Neural Information Processing Systems 19*, 2007.

[23] Martin J. Wainwright, Erik B. Sudderth, and Alan S. Willsky. Tree-based modeling and estimation of gaussian processes on graphs with cycles. In Todd K. Leen, Thomas G. Dietterich, and Volker Tresp, editors, *Advances in Neural Information Processing Systems 13*, pages 661–667. MIT Press, 2001.

[24] Jonathan S. Yedidia, William T. Freeman, and Yair Weiss. Generalized belief propagation. In *Advances in Neural Information Processing Systems 13*. MIT Press, 2001.

[25] J. Zhu, S. Rosset, T. Hastie, and R. Tibshirani. 1-norm support vector machines. In *Proc. NIPS*, 2003.
